# Analysis of SVM with Indefinite Kernels

**Yiming Ying† , Colin Campbell† and Mark Girolami‡**
†Department of Engineering Mathematics, University of Bristol,
Bristol BS8 1TR, United Kingdom
‡Department of Computer Science, University of Glasgow,
S.A.W. Building, G12 8QQ, United Kingdom

## Abstract

The recent introduction of indefinite SVM by Luss and d'Aspremont [15] has effectively demonstrated SVM classification with a non-positive semi-definite kernel (indefinite kernel). This paper studies the properties of the objective function introduced there. In particular, we show that the objective function is continuously differentiable and its gradient can be explicitly computed. Indeed, we further show that its gradient is Lipschitz continuous. The main idea behind our analysis is that the objective function is smoothed by the penalty term, in its saddle (min-max) representation, measuring the distance between the indefinite kernel matrix and the proxy positive semi-definite one. Our elementary result greatly facilitates the application of gradient-based algorithms. Based on our analysis, we further develop Nesterov's smooth optimization approach [17, 18] for indefinite SVM which has an optimal convergence rate for smooth problems. Experiments on various benchmark datasets validate our analysis and demonstrate the efficiency of our proposed algorithms.

## 1 Introduction

Kernel methods [5, 24] such as Support Vector Machines (SVM) have recently attracted much attention due to their good generalization performance and appealing optimization approaches. The basic idea of kernel methods is to map the data into a high dimensional (even infinite-dimensional) feature space through a kernel function. The kernel function over samples forms a similarity kernel matrix which is usually required to be positive semi-definite (PSD). The PSD property of the similarity matrix ensures that the SVM can be efficiently solved by a convex quadratic programming.

However, many potential kernel matrices could be non-positive semi-definite. Such cases are quite common in applications such as the sigmoid kernel [14] for various values of the hyper-parameters, hyperbolic tangent kernels [25], and the protein sequence similarity measures derived from Smith-Waterman and BLAST score [23]. The problem of learning with a non-PSD similarity matrix (indefinite kernel) has recently attracted considerable attention [4, 8, 9, 14, 20, 21, 26]. One widely used method is to convert the indefinite kernel matrix into a PSD one by using the spectrum transformation. The *denoise* method neglects the negative eigenvalues [8, 21], *flip* [8] takes the absolute value of all eigenvalues, *shift* [22] shifts eigenvalues to be positive by adding a positive constant, and the *diffusion* method [11] takes the exponentials of eigenvalues. One can also see [26] for a detailed coverage. However, useful information in the data could be lost in the above spectral transformations since they are separated from the process of training classifiers. In [9], the classification problem with indefinite kernels is regarded as the minimization of the distance between convex hulls in the pseudo-Euclidean space. In [20], general Reproducing Kernel Kreĭn spaces (RKKS) with indefinite kernels are introduced which allows a general representer theorem and regularization formulations.

Luss and d'Aspremont [15] recently proposed a regularized formulation for SVM classification with indefinite kernel. Training a SVM with an indefinite kernel was viewed as a learning the kernel

matrix problem [13] i.e. learning a proxy PSD kernel matrix to approximate the indefinite one. Without realizing that the objective function is differentiable, the authors quadratically smoothed the objective function, and then formulated two approximate algorithms including the projected gradient method and the analytic center cutting plane method.

In this paper we follow the formulation of SVM with indefinite kernels proposed in [15]. We mainly establish the differentiability of the objective function (see its precise definition in equation (3)) and prove that it is, indeed, differentiable with Lipschitz continuous gradient. This elementary result suggests there is no need to smooth the objective function which greatly facilitates the application of gradient-based algorithms. The main idea behind our analysis is from its saddle (min-max) representation which involves a penalty term in the form of Frobenius norm of matrices, measuring the distance between the indefinite kernel matrix and the proxy PSD one. This penalty term can be regarded as a Moreau-Yosida regularization term [12] to smooth out the objective function.

The paper is organized as follows. In Section 2, we review the formulation of indefinite SVM classification presented in [15]. Our main contribution is outlined in Section 3. There, we first show that the objective function of interest is continuously differentiable and its gradient function can be explicitly computed. Indeed, we further show that its gradient is Lipschitz continuous. Based on our analysis, in Section 4 we propose a simplified formulation of the projected gradient method presented in [15] and show that it has a convergence rate of $\mathcal{O}(1/k)$ where $k$ is the iteration number. We further develop Nesterov's smooth optimization approach [17, 18] for indefinite SVM which has an optimal convergence rate of $\mathcal{O}(1/k^2)$ for smooth problems. In Section 5, our analysis and proposed optimization approaches are validated by experiments on various benchmark data sets.

## 2 Indefinite SVM Classification

In this section we review the regularized formulation of indefinite SVM presented in [15]. To this end, we introduce some notation. Let $\mathbb{N}_n = \{1, 2, \ldots, n\}$ for any $n \in \mathbb{N}$ and $\mathcal{S}^n$ be the space of all $n \times n$ symmetric matrices. If $A \in \mathcal{S}^n$ is positive semi-definite, we write it as $A \succeq 0$. The cone of PSD matrices is denoted by $\mathcal{S}^n_+$. For any $A, B \in \mathbb{R}^{n \times n}$, $\langle A, B \rangle_F := \mathbf{Tr}(A^\top B)$ where $\mathbf{Tr}(\cdot)$ denotes the trace of a matrix. Finally, the Frobenius norm over the vector space $\mathcal{S}^n$ is denoted, for any $A \in \mathcal{S}^n$, by $\|A\|_F := (\mathbf{Tr}(A^\top A))^{\frac{1}{2}}$. The standard Euclidean norm and inner product are respectively denoted by $\|\cdot\|$ and $\langle \cdot, \cdot \rangle$.

Let a set of training samples be given by inputs $\mathbf{x} = \{x_i \in \mathbb{R}^d : i \in \mathbb{N}_n\}$ and outputs $\mathbf{y} = \{y_i \in \{\pm 1\} : i \in \mathbb{N}_n\}$. Suppose that $K$ is a positive semi-definite kernel matrix (proxy kernel matrix) on inputs $\mathbf{x}$. Let matrix $Y = \mathrm{diag}(\mathbf{y})$, vector $e$ be an $n$-dimensional vector of all ones and $C$ be a positive trade-off parameter. Then, the dual formulation of 1-norm soft margin SVM [5, 24] is given by

$$\max_\alpha \quad \alpha^\top e - \tfrac{1}{2}\alpha^\top Y K Y \alpha$$
$$\text{s.t.} \quad \alpha^\top \mathbf{y} = 0, 0 \leq \alpha \leq C.$$

Since we assume that $K$ is positive semi-definite, the above problem is a standard convex quadratic program [2] and a global solution can be efficiently obtained by, e.g., the primal-dual interior method. Suppose now we are only given an indefinite kernel matrix $K_0 \in \mathcal{S}^n$. Luss and d'Aspremont [15] proposed the following max-min approach to simultaneously learn a proxy PSD kernel matrix $K$ for the indefinite matrix $K_0$ and the SVM classification:

$$\min_K \max_\alpha \quad \alpha^\top e - \tfrac{1}{2}\alpha^\top Y K Y \alpha + \rho\|K - K_0\|_F^2 \tag{1}$$
$$\text{s.t.} \quad \alpha^\top \mathbf{y} = 0, 0 \leq \alpha \leq C, K \succeq 0.$$

Let $\mathcal{Q}_1 = \{\alpha \in \mathbb{R}^n : \alpha^\top \mathbf{y} = 0, 0 \leq \alpha \leq C\}$ and $\mathcal{L}(\alpha, K) = \alpha^\top e - \tfrac{1}{2}\alpha^\top Y K Y \alpha + \rho\|K - K_0\|_F^2$. By the min-max theorem [2], problem (1) is equivalent to

$$\max_{\alpha \in \mathcal{Q}_1} \min_{K \in \mathcal{S}^n_+} \mathcal{L}(\alpha, K). \tag{2}$$

For simplicity, we refer to the following function defined by

$$f(\alpha) = \min_{K \in \mathcal{S}^n_+} \mathcal{L}(\alpha, K) \tag{3}$$

as the *objective function*. It is obviously concave since $f$ is the minimum of a sequence of concave functions. We also call the associated function $\mathcal{L}(\alpha, K)$ the *saddle representation* of the objective function $f$.

For fixed $\alpha \in \mathcal{Q}_1$, the optimization $K(\alpha) = \arg\min_{K \succeq 0} \mathcal{L}(\alpha, K)$ is equivalent to a projection to the semi-definite cone $\mathcal{S}_+^n$. Indeed, it was shown in [15] that the optimal solution is given by

$$K(\alpha) = (K_0 + Y\alpha\alpha^\top Y/(4\rho))_+ \tag{4}$$

where, for any matrix $A \in \mathcal{S}^n$, the notation $A_+$ denotes the positive part of $A$ by simply setting its negative eigenvalues to zero. The optimal solution $(\alpha^*, K^*) \in \mathcal{Q}_1 \times \mathcal{S}_+^n$ to the above min-max problem is a saddle point of $\mathcal{L}(\alpha, K)$ (see e.g. [2]), i.e. for any $\alpha \in \mathcal{Q}_1, K \in \mathcal{S}_+^n$ there holds $\mathcal{L}(\alpha, K^*) \leq \mathcal{L}(\alpha^*, K^*) \leq \mathcal{L}(\alpha^*, K)$. For a matrix $A \in \mathcal{S}^n$, denote its maximum eigenvalue by $\lambda_{\max}(A)$. The next lemma tells us that the optimal solution $K^*$ belongs to a bounded domain in $\mathcal{S}_+^n$.

**Lemma 1.** *Problem (2) is equivalent to the formulation $\max_{\alpha \in \mathcal{Q}_1} \min_{K \in \mathcal{Q}_2} \mathcal{L}(\alpha, K)$ and the objective function can be defined by*

$$f(\alpha) = \min_{K \in \mathcal{Q}_2} \mathcal{L}(\alpha, K) \tag{5}$$

*where $\mathcal{Q}_2 := \left\{ K \in \mathcal{S}_+^n : \lambda_{max}(K) \leq \lambda_{max}(K_0) + \frac{nC^2}{4\rho} \right\}$.*

*Proof.* By the saddle theorem [2], we have $\mathcal{L}(\alpha^*, K^*) = \min_{K \in \mathcal{Q}_2} \mathcal{L}(\alpha^*, K)$. Combining this with equation (4) yields that $K^* = K(\alpha^*) = (K_0 + Y\alpha^*(\alpha^*)^\top Y/(4\rho))_+$. We can easily see $\lambda_{\max}(K^*) \leq \lambda_{\max}(K_0 + Y\alpha^*(\alpha^*)^\top Y/(4\rho)) \leq \lambda_{\max}(K_0) + \lambda_{\max}(Y\alpha^*(\alpha^*)^\top Y/(4\rho)) \leq \lambda_{\max}(K_0) + \frac{\|\alpha^*\|^2}{4\rho}$, where the second to last inequality uses the property of maximum eigenvalues (e.g. [10, Page 201]) i.e. $\lambda_{\max}(A + B) \leq \lambda_{\max}(A) + \lambda_{\max}(B)$ for any $A, B \in \mathcal{S}^n$. Note that $0 \leq \alpha^* \leq C, \|\alpha^*\|^2 \leq nC^2$. Combining this with the above inequality yields the desired lemma. $\square$

It is worthy of mentioning that it was shown in [18, Theorem 1] that a function $g$ has a Lipschitz continuous gradient if it enjoys a special structure: $g(\alpha) = \min\{\langle A\alpha, K \rangle + \gamma d(K) : K \in \mathcal{Q}\}$ where $Q$ is a closed convex subset in a certain vector space and $d(\cdot)$ is a strongly convex function, and, most importantly, $A$ is a *linear operator*. Since the variable $\alpha$ appeared in a quadratic form, i.e. $\alpha^\top YKY\alpha$, in the objective function defined by (5), it can not be written as the above special form, and hence the theorem there can not be applied to our case.

## 3 Differentiability of the Objective Function

The following lemma outlines a very useful characterization of differentiable properties of the optimal value function [3, Theorem 4.1], essentially due to Danskin [7].

**Lemma 2.** *Let $\mathcal{X}$ be a metric space and $U$ be a normed space. Suppose that for all $x \in \mathcal{X}$ the function $\mathcal{L}(\alpha, \cdot)$ is differentiable, $\mathcal{L}(\alpha, x)$ and $\partial_\alpha \mathcal{L}(\alpha, x)$, the derivative of $\mathcal{L}(\cdot, x)$, are continuous on $\mathcal{X} \times U$ and let $\mathcal{Q}$ be a compact subset of $\mathcal{X}$. Define the optimal value function as $f(\alpha) = \inf_{x \in \mathcal{Q}} \mathcal{L}(\alpha, x)$. The optimal value function is directionally differentiable. Furthermore, if for $\alpha \in U, \mathcal{L}(\alpha, \cdot)$ has a unique minimizer $x(\alpha)$ over $\mathcal{Q}$ then $f$ is differentiable at $\alpha$ and the gradient of $f$ is given by $\nabla f(\alpha) = \partial_\alpha \mathcal{L}(\alpha, x(\alpha))$.*

Applying the above lemma to the objective function $f$ defined by equation (5), we have:

**Theorem 1.** *The objective function $f$ defined by (3) (equivalently by (5)) is differentiable and its gradient is given by*

$$\nabla f(\alpha) = e - Y(K_0 + Y\alpha\alpha^\top Y/(4\rho))_+ Y\alpha. \tag{6}$$

*Proof.* We apply Lemma 2 with $\mathcal{X} = \mathcal{S}^n$ and $\mathcal{Q} = \mathcal{Q}_2 \subseteq \mathcal{S}^n$, $U = \mathcal{Q}_1$ and $x = K$. To this end, we first prove the uniqueness of $K(\alpha)$. Suppose there are two minimizers $K_1, K_2$ for problem $\arg\min_{K \in \mathcal{S}_+^n} \mathcal{L}(\alpha, K)$. By the first order optimality condition, for the minimizer $K_1$, we have that $\langle \partial_K \mathcal{L}(\alpha, K_1), K_2 - K_1 \rangle_F \geq 0$. Considering the minimizer $K_2$, we also have $\langle \partial_K \mathcal{L}(\alpha, K_2), K_1 - K_2 \rangle_F \geq 0$. Noting that $\partial_K \mathcal{L}(\alpha, K) = -\frac{1}{2} Y\alpha\alpha^\top Y + 2\rho(K - K_0)$ and adding the above two first-order optimaility inequalities together, we have $-\|K_2 - K_1\|_F^2 \geq 0$ which means that $K_1 = K_2$, and hence completes the proof of the uniqueness of $K(\alpha)$. Now the desired result follows directly from Lemma 2 by noting that the derivative of $\mathcal{L}$ w.r.t. the first argument $\partial_\alpha \mathcal{L}(\alpha, K) = e - YKY\alpha$. $\square$

Indeed, we can go further to establish the Lipschitz continuity of $\nabla f$ based on the strongly convex property of $\mathcal{L}(\alpha, \cdot)$. To this end, we first establish a useful lemma.

**Lemma 3.** *For any* $\alpha_1, \alpha_2 \in \mathcal{Q}_1$, *there holds* $\|(K_0 + Y\alpha_1\alpha_1^\top Y/(4\rho))_+ - (K_0 + Y\alpha_2\alpha_2^\top Y/(4\rho))_+\|_F \leq (\|\alpha_1\| + \|\alpha_2\|)\|\alpha_1 - \alpha_2\|/(4\rho)$.

*Proof.* Let $\partial_K \mathcal{L}(\alpha, \cdot)$ denote the gradient w.r.t. $K$. Now, consider the minimization problem $\arg\min_{K \in \mathcal{Q}_2} \mathcal{L}(\alpha, K)$. By the first order optimality conditions, for any $K \in \mathcal{Q}_2$ there holds $\langle \partial_K \mathcal{L}(\alpha, K(\alpha)), K - K(\alpha) \rangle_F \geq 0$. Applying the above inequality twice implies that $\langle \partial_K \mathcal{L}(\alpha_1, K(\alpha_1)), K(\alpha_2) - K(\alpha_1) \rangle_F \geq 0$, and $\langle \partial_K \mathcal{L}(\alpha_2, K(\alpha_2)), K(\alpha_1) - K(\alpha_2) \rangle_F \geq 0$. Consequently, $\langle \partial_K \mathcal{L}(\alpha_1, K(\alpha_1)) - \partial_K \mathcal{L}(\alpha_2, K(\alpha_2)), K(\alpha_2) - K(\alpha_1) \rangle_F \geq 0$. Substituting the fact that $\partial_K \mathcal{L}(\alpha, K) = -\frac{1}{2} Y\alpha\alpha^\top Y + 2\rho(K - K_0)$ into the above equation, we have $4\rho\|K(\alpha_1) - K(\alpha_2)\|_F^2 \leq \langle Y(\alpha_2\alpha_2^\top - \alpha_1\alpha_1^\top)Y, K(\alpha_2) - K(\alpha_1) \rangle_F \leq \|Y(\alpha_2\alpha_2^\top - \alpha_1\alpha_1^\top)Y\|_F \|K(\alpha_2) - K(\alpha_1)\|_F$. Consequently,

$$\|K(\alpha_1) - K(\alpha_2)\|_F \leq \frac{\|Y(\alpha_2\alpha_2^\top - \alpha_1\alpha_1^\top)Y\|_F}{4\rho} \leq \frac{\|(\alpha_2\alpha_2^\top - \alpha_1\alpha_1^\top)\|_F}{4\rho} \tag{7}$$

where the last inequality follows from the fact that $Y$ is an orthonormal matrix since $y_i \in \{\pm 1\}$ and $Y = \text{diag}(y_1, \ldots, y_n)$. Note that $\|\alpha_2\alpha_2^\top - \alpha_1\alpha_1^\top\|_F = \|(\alpha_2 - \alpha_1)\alpha_2^\top - \alpha_1(\alpha_1 - \alpha_2)^\top\|_F \leq (\|\alpha_1\| + \|\alpha_2\|)\|\alpha_1 - \alpha_2\|$. Putting this back into inequality (7) completes the proof of the lemma. $\quad\square$

It is interesting to point out that the above lemma can be alternatively established by delicate techniques in matrix analysis. To see this, recall that a *spectral function* $G : \mathcal{S}^n \to \mathcal{S}^n$ is defined by applying a real-valued function $g$ to the eigenvalues of its argument i.e. for any $K \in \mathcal{S}^n$ with eigen-decomposition $K = U\text{diag}(\lambda_1, \ldots, \lambda_n)U^\top$, $G(K) := U\text{diag}(g(\lambda_1), \ldots, g(\lambda_n))U^\top$. The perturbation inequality in matrix analysis [1, Lemma VII.5.5] shows that if $g$ is Lipschitz continuous with Lipschitz constant $\kappa$ then $\|G(K_1) - G(K_2)\|_F \leq \kappa\|K_1 - K_2\|_F, \quad \forall K_1, K_2 \in \mathcal{S}^n$. Applying the above inequality with $g(t) = \max(0, t)$ and $K_1 = K_0 + Y\alpha_1\alpha_1^\top Y/(4\rho)$ and $K_2 = K_0 + Y\alpha_2\alpha_2^\top Y/(4\rho)$ implies equation (7), and hence Lemma 3. However, we prefer the original proof presented for Lemma 3 since it explains more clearly how the strong convexity of the regularization term $\|K - K_0\|_F^2$ plays a critical role in the analysis.

From the above lemma, we can establish the Lipschitz continuity of the gradient of the objective function.

**Theorem 2.** *The gradient of the objective function given by (6) is Lipschitz continuous with Lipschitz constant* $L = \lambda_{max}(K_0) + \frac{nC^2}{\rho}$ *i.e. for any* $\alpha_1, \alpha_2 \in \mathcal{Q}_1$ *the following inequality holds* $\|\nabla f(\alpha_1) - \nabla f(\alpha_2)\| \leq \left[\lambda_{max}(K_0)) + nC^2/\rho\right]\|\alpha_1 - \alpha_2\|$.

*Proof.* For any $\alpha_1, \alpha_2 \in \mathcal{Q}_1$, from representation of $\nabla f$ in Theorem 1 the term $\|\nabla f(\alpha_1) - \nabla f(\alpha_2)\|$ can be bounded by

$$\begin{aligned}
&\left\{\|Y\big[(K_0 + Y\alpha_1\alpha_1^\top Y/(4\rho))_+ - (K_0 + Y\alpha_2\alpha_2^\top Y/(4\rho))_+\big]Y\alpha_1\|\right\} \\
&+ \left\{\|Y(K_0 + Y\alpha_2\alpha_2^\top Y/(4\rho))_+ Y(\alpha_2 - \alpha_1)\|\right\}.
\end{aligned} \tag{8}$$

Now it remains to estimate the two terms within parentheses on the right-hand side of inequality (8). Let's begin with the first one by applying Lemma 3.

$$\begin{aligned}
&\|Y\big((K_0 + Y\alpha_1\alpha_1^\top Y/(4\rho))_+ - (K_0 + Y\alpha_2\alpha_2^\top Y/(4\rho))_+\big)Y\alpha_1\| \\
&\leq \|Y\big((K_0 + Y\alpha_1\alpha_1^\top Y/(4\rho))_+ - (K_0 + Y\alpha_2\alpha_2^\top Y/(4\rho))_+\big)Y\|_F\|\alpha_1\| \\
&\leq \|\big(K_0 + Y\alpha_1\alpha_1^\top Y/(4\rho)\big)_+ - \big(K_0 + Y\alpha_2\alpha_2^\top Y/(4\rho)\big)_+\|_F\|\alpha_1\| \\
&\leq \|\alpha_1\|(\|\alpha_1\| + \|\alpha_2\|)\|\alpha_1 - \alpha_2\|/(4\rho) \leq \frac{nC^2}{2\rho}\|\alpha_1 - \alpha_2\|.
\end{aligned} \tag{9}$$

where the second inequality follows from the fact that $Y$ is an orthonormal matrix. For the second term on the right-hand side of inequality (8), we apply the fact proved in Theorem 1 that $K(\alpha) \in \mathcal{Q}_2$ for any $\alpha \in \mathcal{Q}_1$. Indeed, $\|Y(K_0 + Y\alpha_2\alpha_2^\top Y/(4\rho))_+ Y(\alpha_2 - \alpha_1)\| \leq \lambda_{max}\big(Y(K_0 + Y\alpha_2\alpha_2^\top Y/(4\rho))_+ Y\big)\|\alpha_2 - \alpha_1\| \leq \lambda_{max}\big((K_0 + Y\alpha_2\alpha_2^\top Y/(4\rho))_+\big)\|\alpha_2 - \alpha_1\| \leq \left[\lambda_{max}(K_0) + \frac{nC^2}{4\rho}\right]\|\alpha_1 - \alpha_2\|$. Putting this equation and (9) back into equality (8) completes the proof of Theorem 2. $\quad\square$

| **Simplified Projected Gradient Method (SPGM)** |
|---|
| 1. Choose $\gamma \geq \lambda_{\max}(K_0) + \frac{nC^2}{\rho}$. Let $\varepsilon > 0$, $\alpha_0 \in \mathcal{Q}_1$ be given and set $k = 0$. |
| 2. Compute $\nabla f(\alpha_k) = e - Y\left(K_0 + Y\alpha_k\alpha_k^\top Y/(4\rho)\right)_+ Y\alpha_k$ . |
| 3. $\alpha_{k+1} = \mathcal{P}_{\mathcal{Q}_1}\left(\alpha_k + \nabla f(\alpha_k)/\gamma\right)$ . |
| 4. Set $k \leftarrow k+1$. Go to step 2 until the stopping criterion less than $\varepsilon$. |

Table 1: Pseudo-code of projected gradient method

## 4 Smooth Optimization Algorithms

This section is based on the theoretical analysis above, mainly Theorem 2. We first outline a simplified version of the projected gradient method proposed in [15] and show it has a convergence rate of $\mathcal{O}(1/k)$ where $k$ is the iteration number. We can further develop a smooth optimization approach [17, 18] for indefinite SVM (5). This scheme has an optimal convergence rate $\mathcal{O}(1/k^2)$ for smooth problems which has been applied to various problems, e.g. [6].

### 4.1 Simplified Projected Gradient Method

In [15], the objective function was smoothed by adding a quadratic term (see details in Section 3 there) and then they proposed a projected gradient algorithm to solve this approximation problem. Using the explicit gradient representation in Theorem 1 we formulate its simplified version in Table 1 where the projection $\mathcal{P}_{\mathcal{Q}_1} : \mathbb{R}^n \rightarrow \mathcal{Q}_1$ is defined, for any $\beta \in \mathbb{R}^n$, by

$$\mathcal{P}_{\mathcal{Q}_1}(\beta) = \arg\min_{\alpha \in \mathcal{Q}_1} \|\alpha - \beta\|^2. \tag{10}$$

Indeed, from Theorem 2 we can further obtain the following result by developing the techniques in Sections 2.1.5, 2.2.3 and 2.2.4 of [18].

**Lemma 4.** *Let* $\gamma \geq \left[\lambda_{max}(K_0) + \frac{nC^2}{\rho}\right]$ *and* $\{\alpha_k : k \in \mathbb{N}\}$ *be given by the simplified projected gradient method in Table 1. For any* $\alpha \in \mathcal{Q}_1$, *the following inequality holds* $f(\alpha_{k+1}) \geq f(\alpha) + \gamma\langle\alpha_k - \alpha_{k+1}, \alpha - \alpha_k\rangle + \frac{\gamma}{2}\|\alpha_k - \alpha_{k+1}\|^2$.

*Proof.* We know from Theorem 2 that $\nabla f$ is Lipschitz continuous with Lipschitz constant $L = \lambda_{\max}(K_0) + \frac{nC^2}{\rho}$, then we have $f(\alpha) - f(\alpha_k) - \langle\nabla f(\alpha_k), \alpha - \alpha_k\rangle = \int_0^1 \langle\nabla f(\theta\alpha + (1-\theta)\alpha_k) - \nabla f(\alpha_k), \alpha - \alpha_k\rangle d\theta \geq -L\int_0^1 (1-\theta)\|\alpha - \alpha_k\|^2 d\theta \geq -\frac{\gamma}{2}\|\alpha - \alpha_k\|^2$. Applying this inequality with $\alpha = \alpha_{k+1}$ implies that

$$-f(\alpha_k) - \langle\nabla f(\alpha_k), \alpha_{k+1} - \alpha_k\rangle \geq -f(\alpha_{k+1}) - \frac{\gamma}{2}\|\alpha_{k+1} - \alpha_k\|^2. \tag{11}$$

Let $\phi(\alpha) = -f(\alpha_k) - \nabla f(\alpha_k)(\alpha - \alpha_k) + \frac{\gamma}{2}\|\alpha - \alpha_k\|^2$ which implies that $\alpha_{k+1} = \arg\min_{\alpha \in \mathcal{Q}_1} \phi(\alpha)$. Then, by the first-order optimality condition over $\alpha_{k+1}$ there holds, for any $\alpha \in \mathcal{Q}_1$, $\langle\nabla\phi(\alpha_k), \alpha - \alpha_{k+1}\rangle \geq 0$, i.e. $-\langle\nabla f(\alpha_k), \alpha - \alpha_{k+1}\rangle \geq \gamma\langle\alpha_{k+1} - \alpha_k, \alpha_{k+1} - \alpha\rangle$. Adding this equation and (11) together yields that $-f(\alpha_k) - \langle\nabla f(\alpha_k), \alpha - \alpha_k\rangle \geq -f(\alpha_{k+1}) + \gamma\langle\alpha_k - \alpha_{k+1}, \alpha - \alpha_k\rangle + \frac{\gamma}{2}\|\alpha_k - \alpha_{k+1}\|^2$. Also, since $-f$ is convex, $-f(\alpha) \geq -f(\alpha_k) - \langle\nabla f(\alpha_k), \alpha - \alpha_k\rangle$. Combining this with the above inequality finishes the proof of the lemma. $\square$

**Theorem 3.** *Let* $\gamma \geq \left[\lambda_{max}(K_0) + \frac{nC^2}{\rho}\right]$ *and the iteration sequence* $\{\alpha_k : k \in \mathbb{N}\}$ *be given by the simplified projected gradient method in Table 1. Then, we have that*

$$f(\alpha_{k+1}) \geq f(\alpha_k) + \frac{\gamma}{2}\|\alpha_{k+1} - \alpha_k\|^2, \tag{12}$$

*Moreover,*

$$\max_{\alpha \in \mathcal{Q}_1} f(\alpha) - f(\alpha_k) \leq \frac{\gamma}{2k}\|\alpha_0 - \alpha^*\|^2 \tag{13}$$

*where* $\alpha^*$ *is an optimal solution of problem* $\max_{\alpha \in \mathcal{Q}_1} f(\alpha)$.

| **Nesterov's Smooth Optimization Method (SMM)** |
| --- |
| 1. Let $\varepsilon > 0$, $k = 0$ and initialize $\alpha_0 \in \mathcal{Q}_1$ and let $L = \lambda_{\max}(K_0)) + nC^2/\rho$. |
| 2. Compute $\nabla f(\alpha_k) = e - Y \left(K_0 + Y\alpha_k\alpha_k^\top Y/(4\rho)\right)_+ Y\alpha_k$ . |
| 3. Compute $\gamma_k = \mathcal{P}_{\mathcal{Q}_1}\left(\alpha_k + \nabla f(\alpha_k)/L\right)$ . |
| 4. Compute $\beta_k = \mathcal{P}_{\mathcal{Q}_1}\left(\alpha_0 + \sum_{i=0}^{k}(i+1)\nabla f(\alpha_k)/(2L)\right)$. |
| 5. Set $\alpha_{k+1} = \frac{2}{k+3}\beta_k + \frac{k+1}{k+3}\gamma_k$ . |
| 6. Set $k \leftarrow k + 1$. Go to step 2 until the stopping criterion less than $\varepsilon$. |

Table 2: Pseudo-code of first-order Nesterov's smooth optimization method

*Proof.* Applying Lemma 4 with $\alpha = \alpha_k$ yields inequality (12). To prove inequality (13), we first apply Lemma 4 with $\alpha = \alpha^*$ to get that, for any $i$, $\max_{\alpha \in \mathcal{Q}_1} f(\alpha) - f(\alpha_i) \leq -\gamma\langle\alpha_i - \alpha_{i+1}, \alpha^* - \alpha_i\rangle - \frac{\gamma}{2}\|\alpha_i - \alpha_{i+1}\|^2 = \frac{\gamma}{2}\|\alpha^* - \alpha_i\|^2 - \frac{\gamma}{2}\|\alpha^* - \alpha_{i+1}\|^2$. Adding them over $i$ from 0 and $k - 1$ and also, noting from (12) that $\{\max_{\alpha \in \mathcal{Q}_1} f(\alpha) - f(\alpha_k) : k \in \mathbb{N}\}$ is decreasing, we have that $k\left(\max_{\alpha \in \mathcal{Q}_1} f(\alpha) - f(\alpha_k)\right) \leq \sum_{i=0}^{k-1}\left(\max_{\alpha \in \mathcal{Q}_1} f(\alpha) - f(\alpha_{i+1})\right) \leq \frac{\gamma}{2}\|\alpha^* - \alpha_0\|^2$. This completes the proof of the theorem. $\square$

From the above theorem, the sequence $\{f(\alpha_k) : k \in \mathbb{N}\}$ is monotonically increasing and the iteration complexity of SPGM is $\mathcal{O}(L/\varepsilon)$ for finding an $\varepsilon$-optimal solution.

## 4.2  Nesterov's Smooth Optimization Method

In [18, 17], Nesterov proposed an efficient smooth optimization method for solving convex programming problems of the form
$$\min_{x \in U} g(x)$$
where $g$ is a convex function with Lipschitz continuous gradient, and $U$ is a closed convex set in $\mathbb{R}^n$. Specifically, suppose there exists $L > 0$ such that $\|\nabla g(x) - \nabla g(x')\| \leq L\|x - x'\|, \quad \forall x, x' \in U$. The smooth optimization approach needs to introduce a *proxy-function* $d(x)$ associated with the set $U$. It is assumed to be continuous and strongly convex on $U$ with convexity parameter $\sigma > 0$. Let $x_0 = \arg\min_{x \in U} d(x)$. Without loss of generality, assume that $d(x_0) = 0$. Thus, strong convexity of $d$ means that , for any $x \in U$, $d(x) \geq \frac{1}{2}\sigma\|x - x_0\|^2$. Then, a specific first-order smooth optimization scheme detailed in [18] can be then applied to the function $g$ with convergence rate in $\mathcal{O}(\sqrt{L/\varepsilon})$. The first-order method needs to define a proxy-function associated with $\mathcal{Q}_1$. Here, we define the proxy-function by $d(\alpha) = \frac{1}{2}\|\alpha - \alpha_0\|^2$ with $\alpha_0 \in \mathcal{Q}_1$. The Lipschitz constant of $-f$ is established in Theorem 2 given by $L = \lambda_{\max}(K_0) + nC^2/\rho$. Translating the first-order Nesterov's scheme [18, Section 3] to our problem (5), we can get the smooth optimization algorithm for indefinite SVM, see its pseudo-code in Table 2. One can see [17] for its variants with general step sizes.

The effectiveness of the first-order Nesterov's algorithm largely depends on the Steps 2, 3 and 4 outlined in Table 2. By Theorem 1, the computation of $\nabla f(\alpha_k)$ in Step 2 needs an eigen-decomposition. Steps 3 and 4 are the projection problem (10) by replacing $\beta$ respectively by $\alpha_k + \nabla f(\alpha_k)/L$ and $\alpha_0 + \sum_{i=0}^{k}(i+1)\nabla f(\alpha_i)/(2L)$. The convergence of this optimal method was shown in [18]: $\max_{\alpha \in \mathcal{Q}_1} f(\alpha) - f(\gamma_k) \leq \frac{4L\|\alpha_0 - \alpha^*\|^2}{(k+1)(k+2)}$ where $\alpha^*$ is one of the optimal solutions. It is worthy of pointing out that either $\{f(\alpha_k) : k \in \mathbb{N}\}$ or $\{f(\gamma_k) : k \in \mathbb{N}\}$ may not monotonically increase, however it can be made to monotonically increase by a simple modification of the algorithm [18]. In addition, the above estimation of the Lipschitz constant $L$ could be loose in reality and one could further accelerate the algorithm by using a line search scheme [16].

## 4.3  Related Work and Complexity Discussion

We list the theoretical time complexity of algorithms to run Indefinite SVM. It is worth noting that the number of iterations to reach a target precision of $\varepsilon$ means that $-f(\alpha_k) - \min_{\alpha \in \mathcal{Q}_1} -f(\alpha) = \max_{\alpha \in \mathcal{Q}_1} f(\alpha) - f(\alpha_k) \leq \varepsilon$. However, this does not mean the dual gap as used in [15] is less than $\varepsilon$. In [15], the objective function is smoothed by adding a quadratic term and then they further

proposed a projected gradient algorithm and analytic center cutting plane method (ACCPM)[1]. As proved in Theorem 3, the number of iterations of the projected gradient method is usually $\mathcal{O}(L/\varepsilon)$. In each iteration, the main complexity cost $\mathcal{O}(n^3)$ is from the eigen-decomposition. Hence, the overall complexity of SPGM is $\mathcal{O}(n^3 L/\varepsilon)$. As discussed in [15], ACCPM has an overall complexity is $\mathcal{O}(n^4 \log(1/\varepsilon)^2)$ for finding an $\varepsilon$-optimal solution. However, this method needs to use interior methods at each iteration which would be slow for large scale datasets.

Chen and Ye [4] reformulated indefinite SVM as an appealing semi-infinite quadratically constrained linear programming (SIQCLP) without applying extra smoothing techniques. There, the algorithm iteratively solves a linear programming with a finite number of quadratic constraints. The iteration complexity of semi-infinite linear programming is usually $\mathcal{O}(1/\varepsilon^3)$. In each iteration, one needs to find maximum violation constraints which involves eigen-decomposition of complexity $\mathcal{O}(n^3)$. Hence, the overall complexity is of $\mathcal{O}(n^3/\varepsilon^3)$. The main limitation of this approach is that one needs to save the subset of increasing quadratically constrained conditions indexed by $n \times n$ matrices and iteratively solve a quadratically constrained linear programming (QCLP). The QCLP sub-problem can be solved by general software packages, e.g. Mosek (http://www.mosek.com/), which is generally slow in our experience. This tends to make the algorithm inefficient during the iteration process, although pruning techniques were proposed to avoid too many quadratically constrained conditions.

Based on our theoretical results (Theorem 2), Nesterov's smooth optimization method can be applied. The complexity of this smooth optimization method (SMM) mainly relies on the eigenvalue decomposition on Step 2 listed in Table 2 which costs $\mathcal{O}(n^3)$. Step 3 and 4 are projections onto the convex region $\mathcal{Q}_1$ which costs $\mathcal{O}(n \log n)$ as pointed out in [15]. The first-order smooth optimization approach [17, 18] has iteration complexity $\mathcal{O}(\sqrt{L/\varepsilon})$ for finding an $\varepsilon$-optimal solution. Consequently, the overall complexity is $\mathcal{O}(n^3 \sqrt{L/\varepsilon})$. Hence, from theoretical comparison the complexity of smoothing optimization is better than the simplified projected gradient method (SPGM) and SIQCLP. Compared with ACCPM, SMM has better dependence on the sample number $n$ but with a worse precision i.e. worse dependence on $\varepsilon$.

## 5   Experimental Validation

We run our proposed smooth optimization approach and simplified projected gradient method on various datasets to validate our analysis. The experiments are done on several benchmark data sets from the UCI repository [19] including Sonar, Ionosphere, Heart, Pima Indians Diabetes, Breast Cancer, and USPS with digits 3 and 5. For USPS dataset, we randomly select 600 samples for each digit. All the results reported are based on 10 random training/test partition with ratio $4/1$. In each data split, as in [4] we first generate a Gaussian kernel matrix $K$ with the hyper-parameter determined by cross-validation on the training data using LIBSVM and then construct indefinite matrices by adding a small noisy matrix i.e. $K_0 := K - 0.1\widehat{E}$. Here, the noisy matrix $\widehat{E} = (E + E')/2$ where $E$ is randomly generated by zero mean and identity covariance matrix. For all methods, the parameters $C$ and $\rho$ for Indefinite SVM are tuned by cross-validation and we terminate the algorithm if the relative change of the objective value is less than $10^{-6}$.

In Table 3, we report the average test set accuracy (%) and CPU time (seconds) across different algorithms: smooth optimization method (SMM), simplified projected gradient method (SPGM), analytic center cutting plane method (ACCPM), and semi-infinite quadratically constrained linear programming (SIQCLP). For the QCLP sub-problem in the SIQCLP method, we use Mosek software package (http://www.mosek.com/). We can see that test accuracies are statistically the same across different algorithms, which validates our analysis on the objective function. In particular, we observe that SMM is consistently more efficient than other methods, especially for a large number of training samples. SIQCLP needs much more time since, in each iteration, it needs to solves a quadratically constrained linear programming. In Figure 1, we plot the objective values versus iteration on Sonar and Diabetes for SMM, SPGM, and ACCPM. The SIQCLP approach is not included here since its objective value is not based on the iteration w.r.t. the variable $\alpha$ which does not directly yield an increasing iteration sequence of objective values in contrast to those of the other three algorithms. From Figure 1, we can see that SMM converges faster than SPGM which is consistent with the complexity analysis. The convergence of ACCPM is quite similar to SMM, especially for

| Data | Size | $\lambda_{\min}$ | $\lambda_{\max}$ | SMM | SPGM | ACCPM | SIQCLP |
|---|---|---|---|---|---|---|---|
| Sonar | 208 | $-1.38$ | 21.47 | 76.34% | 76.34% | 75.12% | 76.09% |
| | | | | 0.74s | 5.12s | 3.20s | 244.55s |
| Ionosphere | 351 | -2.08 | 101.34 | 93.14% | 93.43% | 93.54% | 93.54% |
| | | | | 5.47s | 28.93s | 22.73s | 455.81s |
| Heart | 270 | -1.98 | 178.03 | 79.81% | 79.44% | 79.25% | 79.25% |
| | | | | 3.54s | 12.05s | 11.96s | 689.17s |
| Diabetes | 768 | -3.44 | 539.12 | 70.00% | 69.86% | 70.52% | 69.73% |
| | | | | 39.93s | 345.48s | 678.85s | 3134.31s |
| Breast-cancer | 683 | -2.87 | 290.41 | 95.93% | 96.02% | 96.02% | 95.40% |
| | | | | 5.71s | 50.13s | 212.96s | 4610.82s |
| USPS-35 | 1200 | $-3.72$ | 112.65 | 96.33% | 96.33% | 96.04% | 95.54% |
| | | | | 23.22s | 236.00s | 3713.05s | 5199.17s |

Table 3: Average test set accuracy (%) and CPU time in seconds (s) of different algorithms where $\lambda_{\max}(\lambda_{\min})$ denotes the average maximum (minimum) eigenvalues of the indefinite kernel matrix over training samples.

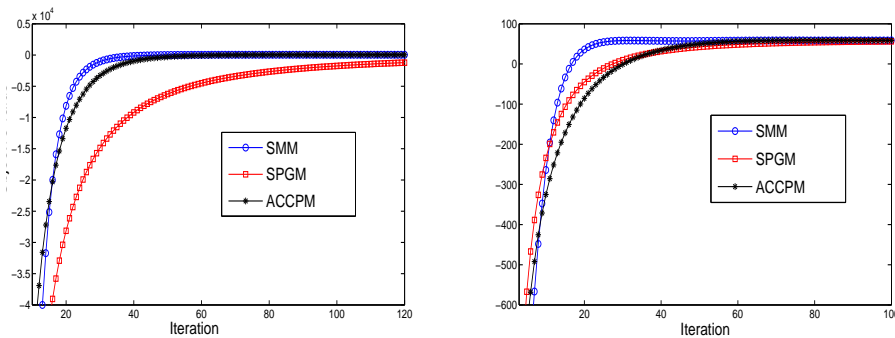

Figure 1: Objective value versus iteration: Sonar (left) and Diabetes (right). Curves: SMM (blue), SPGM (red) and ACCPM (black)

small-sized datasets which coincides with the complexity analysis in Section 4.3 since it generally has a high precision. However, ACCPM needs more time in each iteration than SMM and this observation becomes more apparent for the relatively large datasets shown in the time comparison of Table 3.

# 6 Conclusion

In this paper we analyzed the regularization formulation for training SVM with indefinite kernels proposed by Luss and d'Aspremont [15]. We show that the objective function of interest is continuously differentiable with Lipschitz continuous gradient. Our elementary analysis greatly facilitates the application of gradient-based methods. We formulated a simplified version of the projected gradient method presented in [15] and showed that it has a convergence rate of $\mathcal{O}(1/k)$. We further developed Nesterov's smooth optimization method [17, 18] for Indefinite SVM which has an optimal convergence rate of $\mathcal{O}(1/k^2)$ for smooth problems. Experiments on various datasets validate our analysis and the efficiency of our proposed optimization approach. In future, we are planning to further accelerate the algorithm by using a line search scheme [16]. We are also applying this method to real biological datasets such as protein sequence analysis using sequence alignment measures.

# Acknowledgements

This work is supported by EPSRC grant EP/E027296/1.

## Footnotes

[1]MATLAB codes are available in http://www.princeton.edu/ rluss/IndefiniteSVM.htm

# References

[1] R. Bhatia. *Matrix analysis*. Graduate texts in Mathematics. Springer, 1997.

[2] S. Boyd and L. Vandenberghe. *Convex optimization*. Cambridge University Press, 2004.

[3] J. F. Bonnans and A. Shapiro. Optimization problems with perturbation: A guided tour. *SIAM Review*, **40**: 202–227, 1998.

[4] J. Chen and J. Ye. Training SVM with Indefinite Kernels. *ICML*, 2008.

[5] N. Cristianini and J. Shawe-Taylor. *An introduction to support vector machines and other kernel-based learning methods*. Cambridge University Press, 2000.

[6] A. d'Aspremont, O. Banerjee and L. El Ghaoui. First-order methods for sparse covariance selection. *SIAM Journal on Matrix Analysis and its Applications*, **30**: 56–66, 2007.

[7] J.M. Danskin. *The theory of max-min and its applications to weapons allocation problems*, Springer-Verlag, New York, 1967.

[8] T. Graepel, R. Herbrich, P. Bollmann-Sdorra, and K. Obermayer. Classification on pairwise proximity data. *NIPS*, 1998.

[9] B. Haasdonk. Feature space interpretation of SVMs with indefinite kernels. *IEEE Transactions on Pattern Analysis and Machine Intelligence*, **27**: 482–492, 2005.

[10] R. A. Horn and C. R. Johnson. *Topics in Matrix Analysis*. Cambridge University Press, 1991.

[11] R. I. Kondor and J. Laffferty. Diffusion kernels on graphs and other discrete input spaces. *ICML*, 2002.

[12] C. Lemaréchal and C. Sagastizábal. Practical aspects of the Moreau-Yosida regularization: theoretical preliminaries. *SIAM Journal on Optimization*, **7**: 367–385, 1997.

[13] G. R. G. Lanckriet, N. Cristianini, N., P. L. Bartlett, L. E. Ghaoui, and M. I. Jordan. Learning the kernel matrix with semidefinite programming. *J. of Machine Learning Research*, **5**: 27–72, 2004.

[14] H.-T. Lin and C. J. Lin. A study on sigmoid kernels for SVM and the training of non-psd kernels by smo-type methods. Technical Report, National Taiwan University, 2003.

[15] R. Luss and A. d'Aspremont. Support vector machine classification with indefinite kernels. *NIPS*, 2007.

[16] A. Nemirovski. *Efficient methods in convex programming*. Lecture Notes, 1994.

[17] Y. Nesterov. *Introductory Lectures on Convex Optimization: A Basic Course*. Springer, 2003.

[18] Y. Nesterov. Smooth minimization of non-smooth functions. *Mathematical Programming*, **103**:127–152, 2005.

[19] D. Newman, S. Hettich, C. Blake, and C. Merz. UCI repository of machine learning datasets. 1998.

[20] C. S. Ong, X. Mary, S. Canu, and A. J. Smola. Learning with non-positive kernels. *ICML*, 2004.

[21] E. Pekalska, P. Paclik, and R. P. W. Duin. A generalized kernel approach to dissimilarity-based classification. *J. of Machine Learning Research*, **2**: 175–211, 2002.

[22] V. Roth, J. Laub, M. Kawanabe, and J. M. Buhmann. Optimal cluster preserving embedding of nonmetric proximity data. *IEEE Transactions on Pattern Analysis and Machine Intelligence*, **25**:1540–1551, 2003.

[23] H. Saigo, J.P.Vert and N. Ueda, and T. Akutsu. Protein homology detection using string alignment kernels. *Bioinformatics*, **20**: 1682–1689., 2004.

[24] B. Schölkopf, and A.J. Smola. *Learning with kernels: Support vector machines, regularization, optimization, and beyond*. The MIT Press, 2001.

[25] A. J. Smola, Z. L. Óvári, and R. C. Williamson. Regularization with dot-product kernels. *NIPS*, 2000.

[26] G. Wu, Z. Zhang, and E. Y. Chang. An analysis of transformation on non-positive semidefinite similarity matrix for kernel machines. Technical Report, UCSB, 2005.

